# Source Separation as a By-Product of Regularization

**Sepp Hochreiter**
Fakultät für Informatik
Technische Universität München
80290 München, Germany
hochreit@informatik.tu-muenchen.de

**Jürgen Schmidhuber**
IDSIA
Corso Elvezia 36
6900 Lugano, Switzerland
juergen@idsia.ch

## Abstract

This paper reveals a previously ignored connection between two important fields: regularization and independent component analysis (ICA). We show that at least one representative of a broad class of algorithms (regularizers that reduce network complexity) extracts independent features as a by-product. This algorithm is *Flat Minimum Search* (FMS), a recent general method for finding low-complexity networks with high generalization capability. FMS works by minimizing both training error and required weight precision. According to our theoretical analysis the hidden layer of an FMS-trained autoassociator attempts at coding each input by a sparse code with as few simple features as possible. In experiments the method extracts optimal codes for difficult versions of the "noisy bars" benchmark problem by separating the underlying sources, whereas ICA and PCA fail. Real world images are coded with fewer bits per pixel than by ICA or PCA.

## 1 INTRODUCTION

In the field of unsupervised learning several information-theoretic objective functions (OFs) have been proposed to evaluate the quality of sensory codes. Most OFs focus on properties of the code components — we refer to them as code component-oriented OFs, or COCOFs. Some COCOFs explicitly favor near-factorial, minimally redundant codes of the input data [2, 17, 23, 7, 24] while others favor local codes [22, 3, 15]. Recently there has also been much work on COCOFs encouraging biologically plausible sparse distributed codes [19, 9, 25, 8, 6, 21, 11, 16].

While COCOFs express desirable properties of the code itself they neglect the costs of constructing the code from the data. E.g., coding input data without redun-

dancy may be very expensive in terms of information required to describe the code-generating network, which may need many finely tuned free parameters. We believe that one of sensory coding's objectives should be to reduce the cost of code *generation* through data transformations, and postulate that an important scarce resource is the bits required to describe the mappings that generate and process the codes.

Hence we shift the point of view and focus on the information-theoretic costs of code generation. We use a novel approach to unsupervised learning called "*low-complexity coding* and *decoding*" (LOCOCODE [14]). Without assuming particular goals such as data compression, subsequent classification, etc., but in the spirit of research on minimum description length (MDL), LOCOCODE generates so-called *lococodes* that (1) convey information about the input data, (2) can be computed from the data by a low-complexity mapping (LCM), and (3) can be decoded by an LCM. We will see that by minimizing coding/decoding costs LOCOCODE can yield efficient, robust, noise-tolerant mappings for processing inputs and codes.

**Lococodes through regularizers.** To implement LOCOCODE we apply regularization to an autoassociator (AA) whose hidden layer activations represent the code. The hidden layer is forced to code information about the input data by minimizing training error; the regularizer reduces coding/decoding costs. Our regularizer of choice will be *Flat Minimum Search* (FMS) [13].

## 2   FLAT MINIMUM SEARCH: REVIEW AND ANALYSIS

FMS is a general gradient-based method for finding low-complexity networks with high generalization capability. FMS finds a large region in weight space such that each weight vector from that region has *similar* small error. Such regions are called "flat minima". In MDL terminology, few bits of information are required to pick a weight vector in a "flat" minimum (corresponding to a low-complexity network) — the weights may be given with low precision. FMS automatically prunes weights and units, and reduces output sensitivity with respect to remaining weights and units. Previous FMS applications focused on supervised learning [12, 13].

**Notation.** Let $O, H, I$ denote index sets for output, hidden, and input units, respectively. For $l \in O \cup H$, the activation $y^l$ of unit $l$ is $y^l = f(s_l)$, where $s_l = \sum_m w_{lm} y^m$ is the net input of unit $l$ ($m \in H$ for $l \in O$ and $m \in I$ for $l \in H$), $w_{lm}$ denotes the weight on the connection from unit $m$ to unit $l$, $f$ denotes the activation function, and for $m \in I$, $y^m$ denotes the $m$-th component of an input vector. $W = |(O \times H) \cup (H \times I)|$ is the number of weights.

**Algorithm.** FMS' objective function $E$ features an unconventional error term:

$$B = \sum_{i,j:\ i \in O \cup H} \log \sum_{k \in O} \left( \frac{\partial y^k}{\partial w_{ij}} \right)^2 + W \log \sum_{k \in O} \left( \sum_{i,j:i \in O \cup H} \frac{\left| \frac{\partial y^k}{\partial w_{ij}} \right|}{\sqrt{\sum_{k \in O} \left( \frac{\partial y^k}{\partial w_{ij}} \right)^2}} \right)^2 .$$

$E = E_q + \lambda B$ is minimized by gradient descent, where $E_q$ is the training set mean squared error (MSE), and $\lambda$ a positive "regularization constant" scaling $B$'s influence. Choosing $\lambda$ corresponds to choosing a tolerable error level (there is no *a priori* "optimal" way of doing so). $B$ measures the weight precision (number of bits needed to describe all weights in the net). Given a constant number of output units, FMS can be implemented efficiently, namely, with standard backprop's order of computational complexity [13].

## 2.1 FMS: A Novel Analysis

**Simple basis functions (BFs).** A BF is the function determining the activation of a code component in response to a given input. Minimizing $B$'s term

$$T1 := \sum_{i,j:\, i \in O \cup H} \log \sum_{k \in O} \left( \frac{\partial y^k}{\partial w_{ij}} \right)^2$$

obviously reduces output sensitivity with respect to weights (and therefore units). $T1$ is responsible for pruning weights (and, therefore, units). $T1$ is one reason why low-complexity (or simple) BFs are preferred: weight precision (or complexity) is mainly determined by $\frac{\partial y^k}{\partial w_{ij}}$.

**Sparseness.** Because $T1$ tends to make unit activations decrease to zero it favors sparse codes. But $T1$ also favors a sparse hidden layer in the sense that *few hidden units contribute to producing the output*. $B$'s second term

$$T2 := W \log \sum_{k \in O} \left( \sum_{i,j:\, i \in O \cup H} \frac{\left| \frac{\partial y^k}{\partial w_{ij}} \right|}{\sqrt{\sum_{k \in O} \left( \frac{\partial y^k}{\partial w_{ij}} \right)^2}} \right)^2$$

punishes units with similar influence on the output. We reformulate it:

$$T2 = W \log \left( \sum_{i,j:\, i \in O \cup H} \sum_{u,v:\, u \in O \cup H} \frac{\sum_{k \in O} \left| \frac{\partial y^k}{\partial y^i} \right| \left| \frac{\partial y^k}{\partial y^u} \right|}{\sqrt{\sum_{k \in O} \left( \frac{\partial y^k}{\partial y^i} \right)^2} \sqrt{\sum_{k \in O} \left( \frac{\partial y^k}{\partial y^u} \right)^2}} \right) =$$

$$W \log \left( |O| \, |O \times H|^2 + |I|^2 \sum_{k \in O} \sum_{i \in H} \sum_{u \in H} \frac{\left| \frac{\partial y^k}{\partial y^i} \right| \left| \frac{\partial y^k}{\partial y^u} \right|}{\sqrt{\sum_{k \in O} \left( \frac{\partial y^k}{\partial y^i} \right)^2} \sqrt{\sum_{k \in O} \left( \frac{\partial y^k}{\partial y^u} \right)^2}} \right).$$

See intermediate steps in [14]. We observe: (1) an output unit that is very sensitive with respect to two given hidden units will heavily contribute to $T2$ (compare the numerator in the last term of $T2$). (2) This large contribution can be reduced by making both hidden units have large impact on other output units (see denominator in the last term of $T2$).

**Few separated basis functions.** Hence FMS tries to figure out a way of using (1) as few BFs as possible for determining the activation of each output unit, while simultaneously (2) using the same BFs for determining the activations of as many output units as possible (common BFs). (1) and $T1$ separate the BFs: the force towards simplicity (see $T1$) prevents input information from being channelled through a single BF; the force towards few BFs per output makes them non-redundant. (1) and (2) cause few BFs to determine all outputs.

**Summary.** Collectively $T1$ and $T2$ (which make up $B$) encourage *sparse codes* based on *few separated simple basis functions* producing all outputs. Due to space limitations a more detailed analysis (e.g. linear output activation) had to be left to a TR [14] (on the WWW).

# 3   EXPERIMENTS

We compare LOCOCODE to "independent component analysis" (ICA, e.g., [5, 1, 4, 18]) and "principal component analysis" (PCA, e.g., [20]). ICA is realized by Cardoso's JADE algorithm, which is based on whitening and subsequent joint diagonalization of 4th-order cumulant matrices. To measure the information conveyed by resulting codes we train a standard backprop net on the training set used for code generation. Its inputs are the code components; its task is to reconstruct the original input. The test set consists of 500 off-training set exemplars (in the case of real world images we use a separate test image). *Coding efficiency* is the average number of bits needed to code a test set input pixel. The code components are scaled to the interval $[0, 1]$ and partitioned into discrete intervals. Assuming independence of the code components we estimate the probability of each discrete code value by Monte Carlo sampling on the training set. To obtain the test set codes' bits per pixel (Shannon's optimal value) the average sum of all negative logarithms of code component probabilities is divided by the number of input components. All details necessary for reimplementation are given in [14].

**Noisy bars adapted from [10, 11].** The input is a $5 \times 5$ pixel grid with horizontal and vertical bars at random positions. The task is to extract the independent features (the bars). Each of the 10 possible bars appears with probability $\frac{1}{5}$. In contrast to [10, 11] we allow for bar type mixing — *this makes the task harder.* Bar intensities vary in $[0.1, 0.5]$; input units that see a pixel of a bar are activated correspondingly others adopt activation $-0.5$. We add Gaussian noise with variance 0.05 and mean 0 to each pixel. For ICA and PCA we have to provide information about the number (ten) of independent sources (tests with $n$ assumed sources will be denoted by ICA-$n$ and PCA-$n$). LOCOCODE does not require this — using 25 hidden units (HUs) we expect LOCOCODE to *prune* the 15 superfluous HUs.

**Results.** See Table 1. While the reconstruction errors of all methods are similar, LOCOCODE has the best coding efficiency. 15 of the 25 HUs are indeed automatically pruned: LOCOCODE finds an optimal factorial code which exactly mirrors the pattern generation process. PCA codes and ICA-15 codes, however, are unstructured and dense. While ICA-10 codes are almost sparse and do recognize some sources, the sources are not clearly separated like with LOCOCODE — compare the weight patterns shown in [14].

**Real world images.** Now we use more realistic input data, namely subsections of: 1) the aerial shot of a village, 2) an image of wood cells, and 3) an image of striped piece of wood. Each image has $150 \times 150$ pixels, each taking on one of 256 gray levels. $7 \times 7$ ($5 \times 5$ for village) pixels subsections are randomly chosen as training inputs. Test sets stem from images similar to 1), 2), and 3).

**Results.** For the village image LOCOCODE discovers on-center-off-surround hidden units forming a sparse code. For the other two images LOCOCODE also finds appropriate feature detectors — see weight patterns shown in [14]. Using its compact, low-complexity features it always codes more efficiently than ICA and PCA.

| exp. | input field | meth. | num. comp. | rec. error | code type | bits per pixel: # intervals | | | |
|---|---|---|---|---|---|---|---|---|---|
| | | | | | | 10 | 20 | 50 | 100 |
| bars | 5 × 5 | LOC | 10 | 1.05 | sparse | 0.584 | 0.836 | 1.163 | 1.367 |
| bars | 5 × 5 | ICA | 10 | 1.02 | sparse | 0.811 | 1.086 | 1.446 | 1.678 |
| bars | 5 × 5 | PCA | 10 | 1.03 | dense | 0.796 | 1.062 | 1.418 | 1.655 |
| bars | 5 × 5 | ICA | 15 | 0.71 | dense | 1.189 | 1.604 | 2.142 | 2.502 |
| bars | 5 × 5 | PCA | 15 | 0.72 | dense | 1.174 | 1.584 | 2.108 | 2.469 |
| village | 5 × 5 | LOC | 8 | 1.05 | sparse | 0.436 | 0.622 | 0.895 | 1.068 |
| village | 5 × 5 | ICA | 8 | 1.04 | sparse | 0.520 | 0.710 | 0.978 | 1.165 |
| village | 5 × 5 | PCA | 8 | 1.04 | dense | 0.474 | 0.663 | 0.916 | 1.098 |
| village | 5 × 5 | ICA | 10 | 1.11 | sparse | 0.679 | 0.934 | 1.273 | 1.495 |
| village | 5 × 5 | PCA | 10 | 0.97 | dense | 0.578 | 0.807 | 1.123 | 1.355 |
| village | 7 × 7 | LOC | 10 | 8.29 | sparse | 0.250 | 0.368 | 0.547 | 0.688 |
| village | 7 × 7 | ICA | 10 | 7.90 | dense | 0.318 | 0.463 | 0.652 | 0.796 |
| village | 7 × 7 | PCA | 10 | 9.21 | dense | 0.315 | 0.461 | 0.648 | 0.795 |
| village | 7 × 7 | ICA | 15 | 6.57 | dense | 0.477 | 0.694 | 0.981 | 1.198 |
| village | 7 × 7 | PCA | 15 | 8.03 | dense | 0.474 | 0.690 | 0.972 | 1.189 |
| cell | 7 × 7 | LOC | 11 | 0.840 | sparse | 0.457 | 0.611 | 0.814 | 0.961 |
| cell | 7 × 7 | ICA | 11 | 0.871 | sparse | 0.468 | 0.622 | 0.829 | 0.983 |
| cell | 7 × 7 | PCA | 11 | 0.722 | sparse | 0.452 | 0.610 | 0.811 | 0.960 |
| cell | 7 × 7 | ICA | 15 | 0.360 | sparse | 0.609 | 0.818 | 1.099 | 1.315 |
| cell | 7 × 7 | PCA | 15 | 0.329 | dense | 0.581 | 0.798 | 1.073 | 1.283 |
| piece | 7 × 7 | LOC | 4 | 0.831 | sparse | 0.207 | 0.269 | 0.347 | 0.392 |
| piece | 7 × 7 | ICA | 4 | 0.856 | sparse | 0.207 | 0.276 | 0.352 | 0.400 |
| piece | 7 × 7 | PCA | 4 | 0.830 | sparse | 0.207 | 0.269 | 0.348 | 0.397 |
| piece | 7 × 7 | ICA | 10 | 0.716 | sparse | 0.535 | 0.697 | 0.878 | 1.004 |
| piece | 7 × 7 | PCA | 10 | 0.534 | sparse | 0.448 | 0.590 | 0.775 | 0.908 |

Table 1: *Overview of experiments: name of experiment, input field size, coding method, number of relevant code components (code size), reconstruction error, nature of code observed on the test set. PCA's and ICA's code sizes need to be prewired.* LOCOCODE*'s, however, are found automatically (we always start with 25 HUs). The final 4 columns show the coding efficiency measured in bits per pixel, assuming the real-valued HU activations are partitioned into 10, 20, 50, and 100 discrete intervals.* LOCOCODE *codes most efficiently.*

## 4 CONCLUSION

According to our analysis LOCOCODE attempts to describe single inputs with as few and as simple features as possible. Given the statistical properties of many visual inputs (with few defining features), this typically results in sparse codes. Unlike objective functions of previous methods, however, LOCOCODE's does *not* contain an explicit term enforcing, say, sparse codes — sparseness or independence are not viewed as a good things *a priori*. Instead we focus on the information-theoretic complexity of the mappings used for coding and decoding. The resulting codes typically compromise between conflicting goals. They tend to be sparse and exhibit *low but not minimal* redundancy — if the cost of minimal redundancy is too high.

Our results suggest that LOCOCODE's objective may embody a general principle of unsupervised learning going beyond previous, more specialized ones. We see that there is at least one representative (FMS) of a broad class of algorithms (regularizers that reduce network complexity) which (1) can do optimal feature extraction as a by-product, (2) outperforms traditional ICA and PCA on visual source separation tasks, and (3) unlike ICA does not even need to know the number of independent sources in advance. This reveals an interesting, previously ignored connection be-

tween regularization and ICA, and may represent a first step towards unification of regularization and unsupervised learning.

**More.** Due to space limitations, much additional theoretical and experimental analysis had to be left to a tech report (29 pages, 20 figures) on the WWW: see [14].

**Acknowledgments.** This work was supported by *DFG grant SCHM 942/3-1* and *DFG grant BR 609/10-2* from "Deutsche Forschungsgemeinschaft".

# References

[1] S. Amari, A. Cichocki, and H.H. Yang. A new learning algorithm for blind signal separation. In David S. Touretzky, Michael C. Mozer, and Michael E. Hasselmo, editors, *Advances in Neural Information Processing Systems 8*, pages 757–763. The MIT Press, Cambridge, MA, 1996.

[2] H. B. Barlow, T. P. Kaushal, and G. J. Mitchison. Finding minimum entropy codes. *Neural Computation*, 1(3):412–423, 1989.

[3] H. G. Barrow. Learning receptive fields. In *Proceedings of the IEEE 1st Annual Conference on Neural Networks*, volume IV, pages 115–121. IEEE, 1987.

[4] A. J. Bell and T. J. Sejnowski. An information-maximization approach to blind separation and blind deconvolution. *Neural Computation*, 7(6):1129–1159, 1995.

[5] J.-F. Cardoso and A. Souloumiac. Blind beamforming for non Gaussian signals. *IEE Proceedings-F*, 140(6):362–370, 1993.

[6] P. Dayan and R. Zemel. Competition and multiple cause models. *Neural Computation*, 7:565–579, 1995.

[7] G. Deco and L. Parra. Nonlinear features extraction by unsupervised redundancy reduction with a stochastic neural network. Technical report, Siemens AG, ZFE ST SN 41, 1994.

[8] D. J. Field. What is the goal of sensory coding? *Neural Computation*, 6:559–601, 1994.

[9] P. Földiák and M. P. Young. Sparse coding in the primate cortex. In M. A. Arbib, editor, *The Handbook of Brain Theory and Neural Networks*, pages 895–898. The MIT Press, Cambridge, Massachusetts, 1995.

[10] G. E. Hinton, P. Dayan, B. J. Frey, and R. M. Neal. The wake-sleep algorithm for unsupervised neural networks. *Science*, 268:1158–1161, 1995.

[11] G. E. Hinton and Z. Ghahramani. Generative models for discovering sparse distributed representations. *Philosophical Transactions of the Royal Society* B, 352:1177–1190, 1997.

[12] S. Hochreiter and J. Schmidhuber. Simplifying nets by discovering flat minima. In G. Tesauro, D. S. Touretzky, and T. K. Leen, editors, *Advances in Neural Information Processing Systems 7*, pages 529–536. MIT Press, Cambridge MA, 1995.

[13] S. Hochreiter and J. Schmidhuber. Flat minima. *Neural Computation*, 9(1):1–42, 1997.

[14] S. Hochreiter and J. Schmidhuber. LOCOCODE. Technical Report FKI-222-97, Revised Version, Fakultät für Informatik, Technische Universität München, 1998.

[15] T. Kohonen. *Self-Organization and Associative Memory*. Springer, second ed., 1988.

[16] M. S. Lewicki and B. A. Olshausen. Inferring sparse, overcomplete image codes using an efficient coding framework. In M. I. Jordan, M. J. Kearns, and S. A. Solla, editors, *Advances in Neural Information Processing Systems 10*, 1998. To appear.

[17] R. Linsker. Self-organization in a perceptual network. *IEEE Computer*, 21:105–117, 1988.

[18] L. Molgedey and H. G. Schuster. Separation of independent signals using time-delayed correlations. *Phys. Reviews Letters*, 72(23):3634–3637, 1994.

[19] M. C. Mozer. Discovering discrete distributed representations with iterative competitive learning. In R. P. Lippmann, J. E. Moody, and D. S. Touretzky, editors, *Advances in Neural Information Processing Systems 3*, pages 627–634. San Mateo, CA: Morgan Kaufmann, 1991.

[20] E. Oja. Neural networks, principal components, and subspaces. *International Journal of Neural Systems*, 1(1):61–68, 1989.

[21] B. A. Olshausen and D. J. Field. Emergence of simple-cell receptive field properties by learning a sparse code for natural images. *Nature*, 381(6583):607–609, 1996.

[22] D. E. Rumelhart and D. Zipser. Feature discovery by competitive learning. In *Parallel Distributed Processing*, pages 151–193. MIT Press, 1986.

[23] J. Schmidhuber. Learning factorial codes by predictability minimization. *Neural Computation*, 4(6):863–879, 1992.

[24] S. Watanabe. *Pattern Recognition: Human and Mechanical*. Willey, New York, 1985.

[25] R. S. Zemel and G. E. Hinton. Developing population codes by minimizing description length. In J. D. Cowan, G. Tesauro, and J. Alspector, editors, *Advances in Neural Information Processing Systems 6*, pages 11–18. San Mateo, CA: Morgan Kaufmann, 1994.